# An Empirical Evaluation of Thompson Sampling

**Olivier Chapelle**
Yahoo! Research
Santa Clara, CA
chap@yahoo-inc.com

**Lihong Li**
Yahoo! Research
Santa Clara, CA
lihong@yahoo-inc.com

## Abstract

Thompson sampling is one of oldest heuristic to address the exploration / exploitation trade-off, but it is surprisingly unpopular in the literature. We present here some empirical results using Thompson sampling on simulated and real data, and show that it is highly competitive. And since this heuristic is very easy to implement, we argue that it should be part of the standard baselines to compare against.

## 1 Introduction

Various algorithms have been proposed to solve exploration / exploitation or bandit problems. One of the most popular is *Upper Confidence Bound* or *UCB* [7, 3], for which strong theoretical guarantees on the regret can be proved. Another representative is the Bayes-optimal approach of Gittins [4] that directly maximizes expected cumulative payoffs with respect to a given prior distribution. A less known family of algorithms is the so-called *probability matching*. The idea of this heuristic is old and dates back to [16]. This is the reason why this scheme is also referred to as *Thompson sampling*.

The idea of Thompson sampling is to randomly draw each arm according to its probability of being optimal. In contrast to a full Bayesian method like Gittins index, one can often implement Thompson sampling efficiently. Recent results using Thompson sampling seem promising [5, 6, 14, 12]. The reason why it is not very popular might be because of its lack of theoretical analysis. Only two papers have tried to provide such analysis, but they were only able to prove asymptotic convergence [6, 11].

In this work, we present some empirical results, first on a simulated problem and then on two real-world ones: display advertisement selection and news article recommendation. In all cases, despite its simplicity, Thompson sampling achieves state-of-the-art results, and in some cases significantly outperforms other alternatives like UCB. The findings suggest the necessity to include Thompson sampling as part of the standard baselines to compare against, and to develop finite-time regret bound for this empirically successful algorithm.

## 2 Algorithm

The contextual bandit setting is as follows. At each round we have a context $x$ (optional) and a set of actions $\mathcal{A}$. After choosing an action $a \in \mathcal{A}$, we observe a reward $r$. The goal is to find a policy that selects actions such that the cumulative reward is as large as possible.

Thompson sampling is best understood in a Bayesian setting as follows. The set of past observations $D$ is made of triplets $(x_i, a_i, r_i)$ and are modeled using a parametric likelihood function $P(r|a, x, \theta)$ depending on some parameters $\theta$. Given some prior distribution $P(\theta)$ on these parameters, the posterior distribution of these parameters is given by the Bayes rule, $P(\theta|D) \propto \prod P(r_i|a_i, x_i, \theta)P(\theta)$.

In the realizable case, the reward is a stochastic function of the action, context and the unknown, true parameter $\theta^*$. Ideally, we would like to choose the action maximizing the expected reward, $\max_a \mathbb{E}(r|a, x, \theta^*)$.

Of course, $\theta^*$ is unknown. If we are just interested in maximizing the immediate reward (exploitation), then one should choose the action that maximizes $\mathbb{E}(r|a, x) = \int \mathbb{E}(r|a, x, \theta)P(\theta|D)d\theta$.

But in an exploration / exploitation setting, the probability matching heuristic consists in randomly selecting an action $a$ according to its probability of being optimal. That is, action $a$ is chosen with probability

$$\int \mathbb{I}\left[\mathbb{E}(r|a, x, \theta) = \max_{a'} \mathbb{E}(r|a', x, \theta)\right] P(\theta|D)d\theta,$$

where $\mathbb{I}$ is the indicator function. Note that the integral does not have to be computed explicitly: it suffices to draw a random parameter $\theta$ at each round as explained in Algorithm 1. Implementation of the algorithm is thus efficient and straightforward in most applications.

---
**Algorithm 1** Thompson sampling

---
$\quad D = \emptyset$
$\quad$**for** $t = 1, \ldots, T$ **do**
$\quad\quad$Receive context $x_t$
$\quad\quad$Draw $\theta^t$ according to $P(\theta|D)$
$\quad\quad$Select $a_t = \arg\max_a \mathbb{E}_r(r|x_t, a, \theta^t)$
$\quad\quad$Observe reward $r_t$
$\quad\quad D = D \cup (x_t, a_t, r_t)$
$\quad$**end for**

---

In the standard *$K$-armed Bernoulli bandit*, each action corresponds to the choice of an arm. The reward of the $i$-th arm follows a Bernoulli distribution with mean $\theta_i^*$. It is standard to model the mean reward of each arm using a Beta distribution since it is the conjugate distribution of the binomial distribution. The instantiation of Thompson sampling for the Bernoulli bandit is given in algorithm 2. It is straightforward to adapt the algorithm to the case where different arms use different Beta distributions as their priors.

---
**Algorithm 2** Thompson sampling for the Bernoulli bandit

---
**Require:** $\alpha, \beta$ prior parameters of a Beta distribution
$\quad S_i = 0, F_i = 0, \quad \forall i.$ {Success and failure counters}
$\quad$**for** $t = 1, \ldots, T$ **do**
$\quad\quad$**for** $i = 1, \ldots, K$ **do**
$\quad\quad\quad$Draw $\theta_i$ according to Beta$(S_i + \alpha, F_i + \beta)$.
$\quad\quad$**end for**
$\quad\quad$Draw arm $\hat{i} = \arg\max_i \theta_i$ and observe reward $r$
$\quad\quad$**if** $r = 1$ **then**
$\quad\quad\quad S_{\hat{i}} = S_{\hat{i}} + 1$
$\quad\quad$**else**
$\quad\quad\quad F_{\hat{i}} = F_{\hat{i}} + 1$
$\quad\quad$**end if**
$\quad$**end for**

---

## 3 Simulations

We present some simulation results with Thompson sampling for the Bernoulli bandit problem and compare them to the UCB algorithm. The reward probability of each of the $K$ arms is modeled by a Beta distribution which is updated after an arm is selected (see algorithm 2). The initial prior distribution is Beta(1,1).

There are various variants of the UCB algorithm, but they all have in common that the confidence parameter should increase over time. Specifically, we chose the arm for which the following upper

confidence bound [8, page 278] is maximum:

$$\frac{k}{m} + \sqrt{\frac{2\frac{k}{m}\log\frac{1}{\delta}}{m}} + \frac{2\log\frac{1}{\delta}}{m}, \quad \delta = \sqrt{\frac{1}{t}}, \tag{1}$$

where $m$ is the number of times the arm has been selected and $k$ its total reward. This is a tight upper confidence bound derived from Chernoff's bound.

In this simulation, the best arm has a reward probability of $0.5$ and the $K-1$ other arms have a probability of $0.5 - \varepsilon$. In order to speed up the computations, the parameters are only updated after every 100 iterations. The regret as a function of $T$ for various settings is plotted in figure 1. An asymptotic lower bound has been established in [7] for the regret of a bandit algorithm:

$$R(T) \geq \log(T) \left[ \sum_{i=1}^{K} \frac{p^* - p_i}{D(p_i||p^*)} + o(1) \right], \tag{2}$$

where $p_i$ is the reward probability of the $i$-th arm, $p^* = \max p_i$ and $D$ is the Kullback-Leibler divergence. This lower bound is logarithmic in $T$ with a constant depending on the $p_i$ values. The plots in figure 1 show that the regrets are indeed logarithmic in $T$ (the linear trend on the right hand side) and it turns out that the observed constants (slope of the lines) are close to the optimal constants given by the lower bound (2). Note that the offset of the red curve is irrelevant because of the $o(1)$ term in the lower bound (2). In fact, the red curves were shifted such that they pass through the lower left-hand corner of the plot.

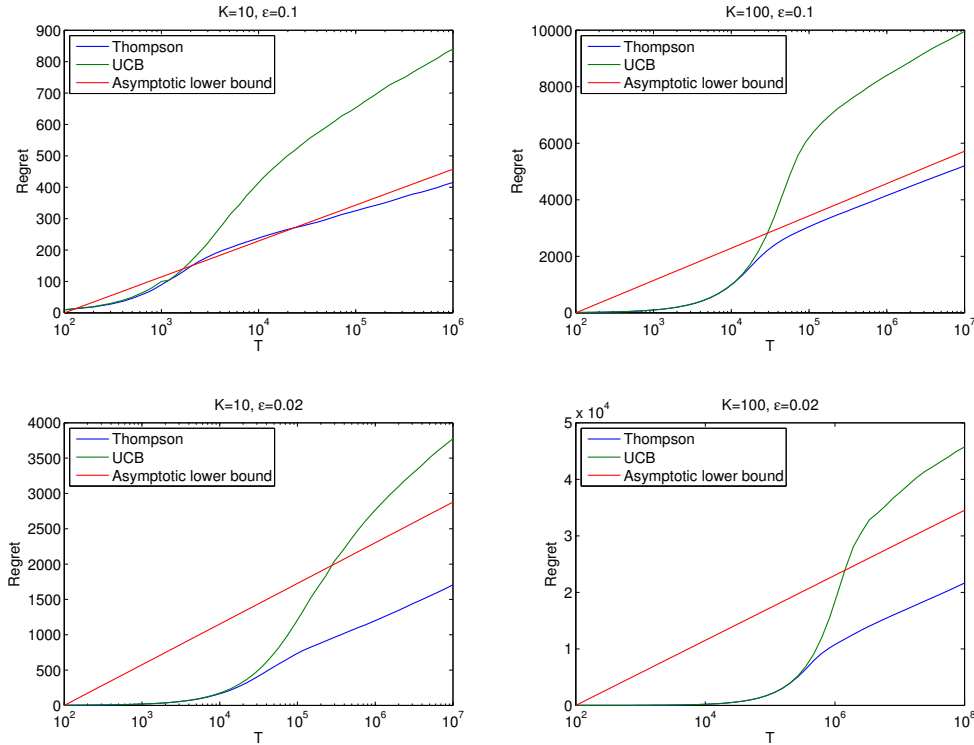

Figure 1: Cumulative regret for $K \in \{10, 100\}$ and $\varepsilon \in \{0.02, 0.1\}$. The plots are averaged over 100 repetitions. The red line is the lower bound (2) shifted such that it goes through the origin.

As with any Bayesian algorithm, one can wonder about the robustness of Thompson sampling to prior mismatch. The results in figure 1 include already some prior mismatch because the Beta prior with parameters (1,1) has a large variance while the true probabilities were selected to be close to

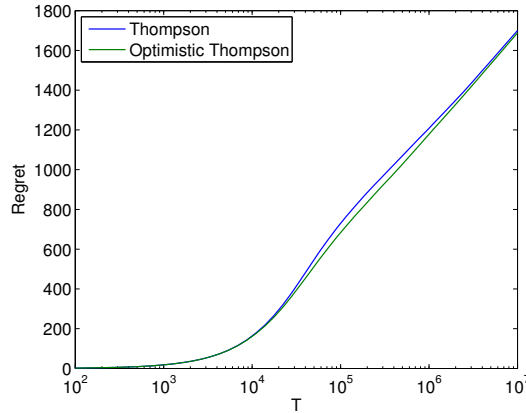

Figure 2: Regret of optimistic Thompson sampling [11] in the same setting as the lower left plot of figure 1.

0.5. We have also done some other simulations (not shown) where there is a mismatch in the prior mean. In particular, when the reward probability of the best arm is 0.1 and the 9 others have a probability of 0.08, Thompson sampling—with the same prior as before—is still better than UCB and is still asymptotically optimal.

We can thus conclude that in these simulations, Thompson sampling is asymptotically optimal and achieves a smaller regret than the popular UCB algorithm. It is important to note that for UCB, the confidence bound (1) is tight; we have tried some other confidence bounds, including the one originally proposed in [3], but they resulted in larger regrets.

**Optimistic Thompson sampling**   The intuition behind UCB and Thompson sampling is that, for the purpose of exploration, it is beneficial to boost the predictions of actions for which we are uncertain. But Thompson sampling modifies the predictions in both directions and there is apparently no benefit in decreasing a prediction. This observation led to a recently proposed algorithm called *Optimistic Bayesian sampling* [11] in which the modified score is never smaller than the mean. More precisely, in algorithm 1, $\mathbb{E}_r(r|x_t, a, \theta^t)$ is replaced by $\max(\mathbb{E}_r(r|x_t, a, \theta^t), \mathbb{E}_{r,\theta|D}(r|x_t, a, \theta))$.

Simulations in [12] showed some gains using this optimistic version of Thompson sampling. We compared in figure 2 the two versions of Thompson sampling in the case $K = 10$ and $\varepsilon = 0.02$. Optimistic Thompson sampling achieves a slightly better regret, but the gain is marginal. A possible explanation is that when the number of arms is large, it is likely that, in standard Thompson sampling, the selected arm has a already a boosted score.

**Posterior reshaping**   Thompson sampling is a heuristic advocating to draw samples from the posterior, but one might consider changing that heuristic to draw samples from a modified distribution. In particular, sharpening the posterior would have the effect of increasing exploitation while widening it would favor exploration. In our simulations, the posterior is a Beta distribution with parameters $a$ and $b$, and we have tried to change it to parameters $a/\alpha$, $b/\alpha$. Doing so does not change the posterior mean, but multiply its variance by a factor close to $\alpha^2$.

Figure 3 shows the average and distribution of regret for different values of $\alpha$. Values of $\alpha$ smaller than 1 decrease the amount of exploration and often result in lower regret. But the price to pay is a higher variance: in some runs, the regret is very large. The average regret is asymptotically not as good as with $\alpha = 1$, but tends to be better in the non-asymptotic regime.

**Impact of delay**   In a real world system, the feedback is typically not processed immediately because of various runtime constraints. Instead it usually arrives in batches over a certain period of time. We now try to quantify the impact of this delay by doing some simulations that mimic the problem of news articles recommendation [9] that will be described in section 5.

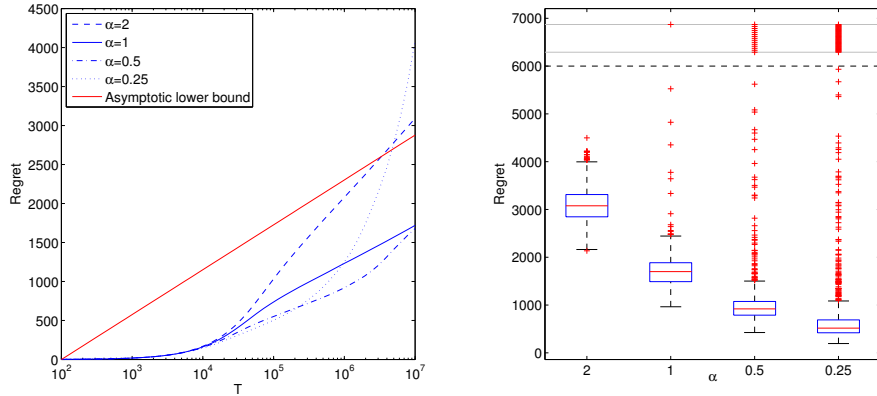

Figure 3: Thompson sampling where the parameters of the Beta posterior distribution have been divided by $\alpha$. The setting is the same as in the lower left plot of figure 1 (1000 repetitions). Left: average regret as a function of $T$. Right: distribution of the regret at $T = 10^7$. Since the outliers can take extreme values, those above 6000 are compressed at the top of the figure.

Table 1: Influence of the delay: regret when the feedback is provided every $\delta$ steps.

| $\delta$ | 1 | 3 | 10 | 32 | 100 | 316 | 1000 |
|---|---|---|---|---|---|---|---|
| UCB | 24,145 | 24,695 | 25,662 | 28,148 | 37,141 | 77,687 | 226,220 |
| TS | 9,105 | 9,199 | 9,049 | 9,451 | 11,550 | 21,594 | 59,256 |
| Ratio | 2.65 | 2.68 | 2.84 | 2.98 | 3.22 | 3.60 | 3.82 |

We consider a dynamic set of 10 items. At a given time, with probability $10^{-3}$ one of the item retires and is replaced by a new one. The true reward probability of a given item is drawn according to a Beta(4,4) distribution. The feedback is received only every $\delta$ time units. Table 1 shows the average regret (over 100 repetitions) of Thompson sampling and UCB at $T = 10^6$. An interesting quantity in this simulation is the relative regret of UCB and Thompson sampling. It appears that Thompson sampling is more robust than UCB when the delay is long. Thompson sampling alleviates the influence of delayed feedback by randomizing over actions; on the other hand, UCB is deterministic and suffers a larger regret in case of a sub-optimal choice.

## 4 Display Advertising

We now consider an online advertising application. Given a user visiting a publisher page, the problem is to select the best advertisement for that user. A key element in this matching problem is the click-through rate (CTR) estimation: what is the probability that a given ad will be clicked given some context (user, page visited)? Indeed, in a cost-per-click (CPC) campaign, the advertiser only pays when his ad gets clicked. This is the reason why it is important to select ads with high CTRs. There is of course a fundamental exploration / exploitation dilemma here: in order to learn the CTR of an ad, it needs to be displayed, leading to a potential loss of short-term revenue. More details on on display advertising and the data used for modeling can be found in [1].

In this paper, we consider standard regularized logistic regression for predicting CTR. There are several features representing the user, page, ad, as well as conjunctions of these features. Some of the features include identifiers of the ad, advertiser, publisher and visited page. These features are hashed [17] and each training sample ends up being represented as sparse binary vector of dimension $2^{24}$.

In our model, the posterior distribution on the weights is approximated by a Gaussian distribution with diagonal covariance matrix. As in the Laplace approximation, the mean of this distribution is the mode of the posterior and the inverse variance of each weight is given by the curvature. The use

of this convenient approximation of the posterior is twofold. It first serves as a prior on the weights to update the model when a new batch of training data becomes available, as described in algorithm 3. And it is also the distribution used in Thompson sampling.

---

**Algorithm 3** Regularized logistic regression with batch updates

**Require:** Regularization parameter $\lambda > 0$.
$\quad m_i = 0$, $q_i = \lambda$. {Each weight $w_i$ has an independent prior $\mathcal{N}(m_i, q_i^{-1})$}
$\quad$ **for** $t = 1, \ldots, T$ **do**
$\quad\quad$ Get a new batch of training data $(\mathbf{x}_j, y_j)$, $\;j = 1, \ldots, n$.
$\quad\quad$ Find $\mathbf{w}$ as the minimizer of: $\quad \dfrac{1}{2} \sum_{i=1}^{d} q_i (w_i - m_i)^2 + \sum_{j=1}^{n} \log(1 + \exp(-y_j \mathbf{w}^\top \mathbf{x}_j))$.
$\quad\quad m_i = w_i$
$\quad\quad q_i = q_i + \sum_{j=1}^{n} x_{ij}^2 p_j (1 - p_j), \;\; p_j = (1 + \exp(-\mathbf{w}^\top \mathbf{x}_j))^{-1}$ {Laplace approximation}
$\quad$ **end for**

---

Evaluating an explore / exploit policy is difficult because we typically do not know the reward of an action that was not chosen. A possible solution, as we shall see in section 5, is to use a *replayer* in which previous, randomized exploration data can be used to produce an *unbiased* offline estimator of the new policy [10]. Unfortunately, their approach cannot be used in our case here because it reduces the effective data size substantially when the number of arms $K$ is large, yielding too high variance in the evaluation results. [15] studies another promising approach using the idea of importance weighting, but the method applies only when the policy is static, which is not the case for online bandit algorithms that constantly adapt to its history.

For the sake of simplicity, therefore, we considered in this section a simulated environment. More precisely, the context and the ads are real, but the clicks are simulated using a weight vector $\mathbf{w}^*$. This weight vector could have been chosen arbitrarily, but it was in fact a perturbed version of some weight vector learned from real clicks. The input feature vectors $\mathbf{x}$ are thus as in the real world setting, but the clicks are artificially generated with probability $P(y = 1|\mathbf{x}) = (1 + \exp(-\mathbf{w}^{*\top} \mathbf{x}))^{-1}$.

About 13,000 contexts, representing a small random subset of the total traffic, are presented every hour to the policy which has to choose an ad among a set of eligible ads. The number of eligible ads for each context depends on numerous constraints set by the advertiser and the publisher. It varies between 5,910 and 1 with a mean of 1,364 and a median of 514 (over a set of 66,373 ads). Note that in this experiment, the number of eligible ads is smaller than what we would observe in live traffic because we restricted the set of advertisers.

The model is updated every hour as described in algorithm 3. A feature vector is constructed for every (context, ad) pair and the policy decides which ad to show. A click for that ad is then generated with probability $(1 + \exp(-\mathbf{w}^{*\top} \mathbf{x}))^{-1}$. This labeled training sample is then used at the end of the hour to update the model. The total number of clicks received during this one hour period is the reward. But in order to eliminate unnecessary variance in the estimation, we instead computed the expectation of that number since the click probabilities are known.

Several explore / exploit strategies are compared; they only differ in the way the ads are selected; all the rest, including the model updates, is identical as described in algorithm 3. These strategies are:

**Thompson sampling** This is algorithm 1 where each weight is drawn independently according to its Gaussian posterior approximation $\mathcal{N}(m_i, q_i^{-1})$ (see algorithm 3). As in section 3, we also consider a variant in which the standard deviations $q_i^{-1/2}$ are first multiplied by a factor $\alpha \in \{0.25, 0.5\}$. This favors exploitation over exploration.

**LinUCB** This is an extension of the UCB algorithm to the parametric case [9]. It selects the ad based on mean and standard deviation. It also has a factor $\alpha$ to control the exploration / exploitation trade-off. More precisely, LinUCB selects the ad for which $\sum_{i=1}^{d} m_i x_i + \alpha \sqrt{\sum_{i=1}^{d} q_i^{-1} x_i^2}$ is maximum.

**Exploit-only** Select the ad with the highest mean.

**Random** Select the ad uniformly at random.

Table 2: CTR regrets on the display advertising data.

| Method | TS | | | LinUCB | | | $\varepsilon$-greedy | | | Exploit | Random |
|---|---|---|---|---|---|---|---|---|---|---|---|
| Parameter | 0.25 | 0.5 | 1 | 0.5 | 1 | 2 | 0.005 | 0.01 | 0.02 | | |
| Regret (%) | 4.45 | 3.72 | 3.81 | 4.99 | 4.22 | 4.14 | 5.05 | 4.98 | 5.22 | 5.00 | 31.95 |

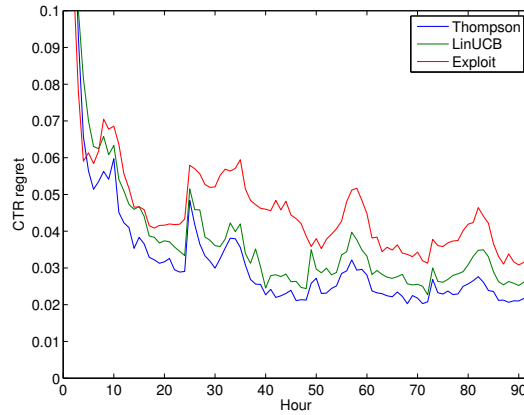

Figure 4: CTR regret over the 4 days test period for 3 algorithms: Thompson sampling with $\alpha = 0.5$, LinUCB with $\alpha = 2$, Exploit-only. The regret in the first hour is large, around 0.3, because the algorithms predict randomly (no initial model provided).

$\varepsilon$-**greedy** Mix between exploitation and random: with $\varepsilon$ probability, select a random ad; otherwise, select the one with the highest mean.

**Results** A preliminary result is about the quality of the variance prediction. The diagonal Gaussian approximation of the posterior does not seem to harm the variance predictions. In particular, they are very well calibrated: when constructing a 95% confidence interval for CTR, the true CTR is in this interval 95.1% of the time.

The regrets of the different explore / exploit strategies can be found in table 2. Thompson sampling achieves the best regret and interestingly the modified version with $\alpha = 0.5$ gives slightly better results than the standard version ($\alpha = 1$). This confirms the results of the previous section (figure 3) where $\alpha < 1$ yielded better regrets in the non-asymptotic regime.

Exploit-only does pretty well, at least compared to random selection. This seems at first a bit surprising given that the system has no prior knowledge about the CTRs. A possible explanation is that the change in context induces some exploration, as noted in [13]. Also, the fact that exploit-only is so much better than random might explain why $\varepsilon$-greedy does not beat it: whenever this strategy chooses a random action, it suffers a large regret in average which is not compensated by its exploration benefit.

Finally figure 4 shows the regret of three algorithms across time. As expected, the regret has a decreasing trend over time.

## 5 News Article Recommendation

In this section, we consider another application of Thompson sampling in personalized news article recommendation on Yahoo! front page [2, 9]. Each time a user visits the portal, a news article out of a small pool of hand-picked candidates is recommended. The candidate pool is dynamic: old articles may retire and new articles may be added in. The average size of the pool is around 20. The goal is to choose the most interesting article to users, or formally, maximize the total number of clicks on the recommended articles. In this case, we treat articles as arms, and define the payoff to be 1 if the article is clicked on and 0 otherwise. Therefore, the average per-trial payoff of a policy is its overall CTR.

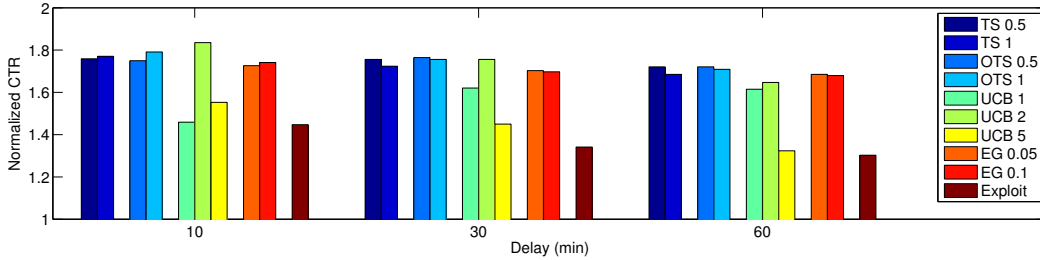

Figure 5: Normalized CTRs of various algorithm on the news article recommendation data with different update delays: $\{10, 30, 60\}$ minutes. The normalization is with respect to a random baseline.

Each user was associated with a binary raw feature vector of over 1000 dimension, which indicates information of the user like age, gender, geographical location, behavioral targeting, etc. These features are typically sparse, so using them directly makes learning more difficult and is computationally expensive. One can find lower dimension feature subspace by, say, following previous practice [9]. Here, we adopted the simpler principal component analysis (PCA), which did not appear to affect the bandit algorithms much in our experience. In particular, we performed a PCA and projected the raw user feature onto the first 20 principal components. Finally, a constant feature 1 is appended, so that the final user feature contains 21 components. The constant feature serves as the bias term in the CTR model described next.

We use logistic regression, as in Algorithm 3, to model article CTRs: given a user feature vector $\mathbf{x} \in \Re^{21}$, the probability of click on an article $a$ is $(1 + \exp(-\mathbf{x}^\top \mathbf{w}_a))^{-1}$ for some weight vector $\mathbf{w}_a \in \Re^{21}$ to be learned. The same parameter algorithm and exploration heuristics are applied as in the previous section. Note that we have a different weight vector for each article, which is affordable as the numbers of articles and features are both small. Furthermore, given the size of data, we have not found article features to be helpful. Indeed, it is shown in our previous work [9, Figure 5] that article features are helpful in this domain only when data are highly sparse.

Given the small size of candidate pool, we adopt the unbiased offline evaluation method of [10] to compare various bandit algorithms. In particular, we collected randomized serving events for a random fraction of user visits; in other words, these random users were recommended an article chosen uniformly from the candidate pool. From 7 days in June 2009, over 34M randomized serving events were obtained.

As in section 3, we varied the update delay to study how various algorithms degrade. Three values were tried: 10, 30, and 60 minutes. Figure 5 summarizes the overall CTRs of four families of algorithm together with the exploit-only baseline. As in the previous section, (optimistic) Thompson sampling appears competitive across all delays. While the deterministic UCB works well with short delay, its performance drops significantly as the delay increases. In contrast, randomized algorithms are more robust to delay, and when there is a one-hour delay, (optimistic) Thompson sampling is significant better than others (given the size of our data).

## 6 Conclusion

The extensive experimental evaluation carried out in this paper reveals that Thompson sampling is a very effective heuristic for addressing the exploration / exploitation trade-off. In its simplest form, it does not have any parameter to tune, but our results show that tweaking the posterior to reduce exploration can be beneficial. In any case, Thompson sampling is very easy to implement and should thus be considered as a standard baseline. Also, since it is a randomized algorithm, it is robust in the case of delayed feedback.

Future work includes of course, a theoretical analysis of its finite-time regret. The benefit of this analysis would be twofold. First, it would hopefully contribute to make Thompson sampling as popular as other algorithms for which regret bounds exist. Second, it could provide guidance on tweaking the posterior in order to achieve a smaller regret.

# References

[1] D. Agarwal, R. Agrawal, R. Khanna, and N. Kota. Estimating rates of rare events with multiple hierarchies through scalable log-linear models. In *Proceedings of the 16th ACM SIGKDD international conference on Knowledge discovery and data mining*, pages 213–222, 2010.

[2] Deepak Agarwal, Bee-Chung Chen, Pradheep Elango, Nitin Motgi, Seung-Taek Park, Raghu Ramakrishnan, Scott Roy, and Joe Zachariah. Online models for content optimization. In *Advances in Neural Information Processing Systems 21*, pages 17–24, 2008.

[3] P. Auer, N. Cesa-Bianchi, and P. Fischer. Finite-time analysis of the multiarmed bandit problem. *Machine learning*, 47(2):235–256, 2002.

[4] John C. Gittins. *Multi-armed Bandit Allocation Indices*. Wiley Interscience Series in Systems and Optimization. John Wiley & Sons Inc, 1989.

[5] Thore Graepel, Joaquin Quinonero Candela, Thomas Borchert, and Ralf Herbrich. Web-scale Bayesian click-through rate prediction for sponsored search advertising in Microsoft's Bing search engine. In *Proceedings of the Twenty-Seventh International Conference on Machine Learning (ICML-10)*, pages 13–20, 2010.

[6] O.-C. Granmo. Solving two-armed bernoulli bandit problems using a bayesian learning automaton. *International Journal of Intelligent Computing and Cybernetics*, 3(2):207–234, 2010.

[7] T.L. Lai and H. Robbins. Asymptotically efficient adaptive allocation rules. *Advances in applied mathematics*, 6:4–22, 1985.

[8] J. Langford. Tutorial on practical prediction theory for classification. *Journal of Machine Learning Research*, 6(1):273–306, 2005.

[9] L. Li, W. Chu, J. Langford, and R. E. Schapire. A contextual-bandit approach to personalized news article recommendation. In *Proceedings of the 19th international conference on World wide web*, pages 661–670, 2010.

[10] L. Li, W. Chu, J. Langford, and X. Wang. Unbiased offline evaluation of contextual-bandit-based news article recommendation algorithms. In *Proceedings of the fourth ACM international conference on Web search and data mining*, pages 297–306, 2011.

[11] Benedict C. May, Nathan Korda, Anthony Lee, and David S. Leslie. Optimistic Bayesian sampling in contextual-bandit problems. Technical Report 11:01, Statistics Group, Department of Mathematics, University of Bristol, 2011. Submitted to the Annals of Applied Probability.

[12] Benedict C. May and David S. Leslie. Simulation studies in optimistic Bayesian sampling in contextual-bandit problems. Technical Report 11:02, Statistics Group, Department of Mathematics, University of Bristol, 2011.

[13] J. Sarkar. One-armed bandit problems with covariates. *The Annals of Statistics*, 19(4):1978–2002, 1991.

[14] S. Scott. A modern bayesian look at the multi-armed bandit. *Applied Stochastic Models in Business and Industry*, 26:639–658, 2010.

[15] Alexander L. Strehl, John Langford, Lihong Li, and Sham M. Kakade. Learning from logged implicit exploration data. In *Advances in Neural Information Processing Systems 23 (NIPS-10)*, pages 2217–2225, 2011.

[16] William R. Thompson. On the likelihood that one unknown probability exceeds another in view of the evidence of two samples. *Biometrika*, 25(3–4):285–294, 1933.

[17] K. Weinberger, A. Dasgupta, J. Attenberg, J. Langford, and A. Smola. Feature hashing for large scale multitask learning. In *ICML*, 2009.

